# Dimensionality Dependent PAC-Bayes Margin Bound

**Chi Jin**
Key Laboratory of Machine Perception, MOE
School of Physics
Peking University
chijin06@gmail.com

**Liwei Wang**
Key Laboratory of Machine Perception, MOE
School of EECS
Peking University
wanglw@cis.pku.edu.cn

## Abstract

Margin is one of the most important concepts in machine learning. Previous margin bounds, both for SVM and for boosting, are dimensionality independent. A major advantage of this dimensionality independency is that it can explain the excellent performance of SVM whose feature spaces are often of high or infinite dimension. In this paper we address the problem whether such dimensionality independency is intrinsic for the margin bounds. We prove a dimensionality dependent PAC-Bayes margin bound. The bound is monotone increasing with respect to the dimension when keeping all other factors fixed. We show that our bound is strictly sharper than a previously well-known PAC-Bayes margin bound if the feature space is of finite dimension; and the two bounds tend to be equivalent as the dimension goes to infinity. In addition, we show that the VC bound for linear classifiers can be recovered from our bound under mild conditions. We conduct extensive experiments on benchmark datasets and find that the new bound is useful for model selection and is usually significantly sharper than the dimensionality independent PAC-Bayes margin bound as well as the VC bound for linear classifiers.

## 1  Introduction

Linear classifiers, including SVM and boosting, play an important role in machine learning. A central concept in the generalization analysis of linear classifiers is margin. There have been extensive works on bounding the generalization errors of SVM and boosting in terms of margins (with various definitions such $l_2$, $l_1$, soft, hard, average, minimum, etc.)

In 1970's Vapnik pointed out that large margin can imply good generalization. Using the fat-shattering dimension, Shawe-Taylor et al. [1] proved a margin bound for linear classifiers. This bound was improved and simplified in a series of works [2, 3, 4, 5] mainly based on the PAC-Bayes theory [6] which was developed originally for stochastic classifiers. (See Section 2 for a brief review of the PAC-Bayes theory and the PAC-Bayes margin bounds.) All these bounds state that if a linear classifier in the feature space induces large margins for most of the training examples, then it has a small generalization error bound independent of the dimensionality of the feature space.

The ($l_1$) margin has also been extensively studied for boosting to explain its generalization ability. Schapire et al. [7] proved a margin bound for the generalization error of voting classifiers. The bound is independent of the number of base classifiers combined in the voting classifier[1]. This margin bound was greatly improved in [8, 9] using (local) Rademacher complexities. There also exist improved margin bounds for boosting from the viewpoint of PAC-Bayes theory [10], the diversity of base classifiers [11], and different definition of margins [12, 13].

The aforementioned margin bounds are all dimensionality independent. That is, the bounds are solely characterized by the margins on the training data and do not depend on the dimension of feature space. A major advantage of such dimensionality independent margin bounds is that they can explain the generalization ability of SVM and boosting whose feature spaces have high or infinite dimension, in which case the standard VC bound becomes trivial.

Although very successful in bounding the generalization error, a natural question is whether this dimensionality independency is intrinsic for margin bounds. In this paper we explore this problem. Building upon the PAC-Bayes theory, we prove a dimensionality *dependent* margin bound. This bound is monotone increasing with respect to the dimension when keeping all other factors fixed. Comparing with the PAC-Bayes margin bound of Langford [4], the new bound is strictly sharper when the feature space is of finite dimension; and the two bounds tend to be equal as the dimension goes to infinity.

We conduct extensive experiments on benchmark datasets. The experimental results show that the new bound is significantly sharper than the dimensionality independent PAC-Bayes margin bound as well as the VC bound for linear classifiers on relatively large datasets. The bound is also found useful for model selection.

The rest of this paper is organized as follows. Section 2 contains a brief review of the PAC-Bayes theory and the dimensionality independent PAC-Bayes margin bound. In Section 3 we give the dimensionality dependent PAC-Bayes margin bound and further improvements. We provide the experimental results in Section 4, and conclude in Section 5. Due to the space limit, all the proofs are given in the supplementary material.

## 2 Background

Let $\mathcal{X}$ be the instance space or generally the feature space. In this paper we always assume $\mathcal{X} = \mathbb{R}^d$. We consider binary classification problems and let $\mathcal{Y} = \{-1, 1\}$. Examples are drawn independently according to an underlying distribution $\mathcal{D}$ over $\mathcal{X} \times \mathcal{Y}$. Let $P_{\mathcal{D}}(A(\mathbf{x}, y))$ denote the probability of event $A$ when an example $(\mathbf{x}, y)$ is chosen according to $\mathcal{D}$. Let $\mathcal{S}$ denote a training set of $n$ i.i.d. examples. We denote by $P_{\mathcal{S}}(A(\mathbf{x}, y))$ the probability of event $A$ when an example $(\mathbf{x}, y)$ is chosen at random from $\mathcal{S}$. Similarly we denote by $E_{\mathcal{D}}$ and $E_{\mathcal{S}}$ the corresponding expectations. If $c$ is a classifier, then we denote by $er_{\mathcal{D}}(c) = P_{\mathcal{D}}(y \neq c(\mathbf{x}))$ the generalization error of $c$, and let $er_{\mathcal{S}}(c) = P_{\mathcal{S}}(y \neq c(\mathbf{x}))$ be the empirical error.

An important type of classifiers studied in this paper is stochastic classifiers. Let $\mathcal{C}$ be a set of classifiers, and let $Q$ be a probability distribution of classifiers on $\mathcal{C}$. A stochastic classifier defined by $Q$ randomly selects $c \in \mathcal{C}$ according to $Q$. When clear from the context, we often denote by $er_{\mathcal{D}}(Q)$ and $er_{\mathcal{S}}(Q)$ the generalization and empirical error of the stochastic classifier $Q$ respectively. That is,

$$er_{\mathcal{D}}(Q) = E_{c \sim Q}[er_{\mathcal{D}}(c)]; \qquad er_{\mathcal{S}}(Q) = E_{c \sim Q}[er_{\mathcal{S}}(c)]$$

A probability distribution $Q$ of classifiers also defines a deterministic classifier—the voting classifier, which we denote by $v_Q$. For $\mathbf{x} \in \mathcal{X}$

$$v_Q(\mathbf{x}) = \text{sgn}[E_{c \sim Q} c(\mathbf{x})].$$

In this paper we always consider homogeneous linear classifiers[2], or stochastic classifiers whose distribution is over homogeneous linear classifiers. Let $\mathcal{X} = \mathbb{R}^d$. For any $\mathbf{w} \in \mathbb{R}^d$, the linear classifier $c_{\mathbf{w}}$ is defined as $c_{\mathbf{w}}(\cdot) = \text{sgn}[< \mathbf{w}, \cdot >]$. When we consider a probability distribution over all homogeneous linear classifiers $c_{\mathbf{w}}$ in $\mathbb{R}^d$, we can equivalently consider a distribution of $\mathbf{w} \in \mathbb{R}^d$.

The work in this paper is based on the PAC-Bayes theory. PAC-Bayes theory is a beautiful generalization of the classical PAC theory to the setting of Bayes learning. It gives generalization error bounds for stochastic classifiers. The PAC-Bayes theorem was first proposed by McAllester [6]. The following elegant version is due to Langford [4].

**Theorem 2.1.** *Let $P$, $Q$ denote probability distributions of classifiers. For any $P$ and any $\delta \in (0, 1)$, with probability $1 - \delta$ over the random draw of $n$ training examples*

$$kl\left(er_{\mathcal{S}}(Q) \,||\, er_{\mathcal{D}}(Q)\right) \leq \frac{KL(Q||P) + \ln \frac{n+1}{\delta}}{n} \tag{1}$$

*holds simultaneously for all distributions $Q$. Here $KL(Q||P)$ is the Kullback-Leibler divergence of distributions $Q$ and $P$; $kl(a||b)$ for $a, b \in [0, 1]$ is the Bernoulli KL divergence defined as $kl(a||b) = a \log \frac{a}{b} + (1 - a) \log \frac{1-a}{1-b}$.*

The above PAC-Bayes theorem states that if a stochastic classifier, whose distribution $Q$ is close (in the sense of KL divergence) to the fixed prior $P$, has a small training error, then its generalization error is small.

PAC-Bayes theory has been improved and generalized in a series of works [5, 14]. For important recent results please referred to [14]. [15] generalizes the KL divergence in the PAC-Bayes theorem to arbitrary convex functions. [15, 16, 17, 18, 19] utilize improved PAC-Bayes bounds to develop learning algorithms and perform model selections.

Very interestingly, it is shown in [2] that one can derive a margin bound for linear classifiers (including SVM) from the PAC-Bayes theorem quite easily. It is much simpler and slightly tighter than previous margin bounds for SVM [1, 20]. The following simplified and refined version can be found in [4].

**Theorem 2.2** ([4]). *Let $\mathcal{X} = \mathbb{R}^d$. Let $Q(\mu, \hat{\boldsymbol{w}})$ ($\mu > 0$, $\hat{\boldsymbol{w}} \in \mathbb{R}^d$, $\|\hat{\boldsymbol{w}}\| = 1$) denote the distribution of homogeneous linear classifiers $c_{\boldsymbol{w}}$, where $\boldsymbol{w} \sim \mathcal{N}(\mu\hat{\boldsymbol{w}}, I)$. For any $\delta \in (0, 1)$, with probability $1 - \delta$ over the random draw of $n$ training examples*

$$kl\left(er_{\mathcal{S}}(Q(\mu, \hat{\boldsymbol{w}})) \,||\, er_{\mathcal{D}}(Q(\mu, \hat{\boldsymbol{w}}))\right) \leq \frac{\frac{\mu^2}{2} + \ln \frac{n+1}{\delta}}{n} \tag{2}$$

*holds simultaneously for all $\mu > 0$ and all $\hat{\boldsymbol{w}} \in \mathbb{R}^d$ with $\|\hat{\boldsymbol{w}}\| = 1$. In addition, the empirical error of the stochastic classifier can be written as*

$$er_{\mathcal{S}}(Q(\mu, \hat{\boldsymbol{w}})) = E_{\mathcal{S}}\overline{\Phi}(\mu\gamma(\hat{\boldsymbol{w}}; \boldsymbol{x}, y)), \tag{3}$$

*where $\gamma(\hat{\boldsymbol{w}}; \boldsymbol{x}, y) = y \frac{<\hat{\boldsymbol{w}}, \boldsymbol{x}>}{\|\boldsymbol{x}\|}$ is the margin of $(\boldsymbol{x}, y)$ with respect to the unit vector $\hat{\boldsymbol{w}}$; and*

$$\overline{\Phi}(t) = 1 - \Phi(t) = \int_t^\infty \frac{1}{\sqrt{2\pi}} e^{-\tau^2/2} d\tau \tag{4}$$

*is the probability of the upper tail of Gaussian distribution.*

According to Theorem 2.2, if there is a linear classifier $\hat{\mathbf{w}} \in \mathbb{R}^d$ inducing large margins for most training examples, i.e., $\gamma(\hat{\mathbf{w}}; \mathbf{x}, y)$ is large for most $(\mathbf{x}, y)$, then choosing a relatively small $\mu$ would yield a small $er_{\mathcal{S}}(Q(\mu, \hat{\mathbf{w}}))$ and in turn a small upper bound for the generalization error of the stochastic classifier $Q(\mu, \hat{\mathbf{w}})$. Note that this bound does not depend on the dimensionality $d$. In fact almost all previously known margin bounds are dimensionality independent[3].

PAC-Bayes theory only provides bounds for stochastic classifiers. In practice however, users often prefer deterministic classifiers. There is a close relation between the error of a stochastic classifier defined by distribution $Q$ and the error of the deterministic voting classifier $v_Q$. The following simple result is well-known.

**Proposition 2.3.** *Let $v_Q$ be the voting classifier defined by distribution $Q$. That is, $v_Q(\cdot) = \text{sgn}[E_{c \sim Q} c(\cdot)]$. Then for any $Q$*

$$er_{\mathcal{D}}(v_Q) \leq 2 \, er_{\mathcal{D}}(Q). \tag{5}$$

Combining Theorem 2.2 and Proposition 2.3, one can upper bound the generalization error of the voting classifier $v_Q$ associated with $Q(\mu, \hat{\mathbf{w}})$ given in Theorem 2.2. In fact, it is easy to see that $v_Q = c_{\hat{\mathbf{w}}}$, the voting classifier is exactly the linear classifier $\hat{\mathbf{w}}$. Thus

$$er_{\mathcal{D}}(c_{\hat{\mathbf{w}}}) \leq 2er_{\mathcal{D}}(Q(\mu, \hat{\mathbf{w}})). \tag{6}$$

From Theorem 2.2, Proposition 2.3 and (6), we have that with probability $1-\delta$ the following margin bound holds for all classifiers $c_{\hat{\mathbf{w}}}$ with $\hat{\mathbf{w}} \in R^d$, $\|\hat{\mathbf{w}}\| = 1$ and all $\mu > 0$:

$$kl \left( er_{\mathcal{S}}(Q(\mu, \hat{\mathbf{w}})) \,\|\, \frac{er_{\mathcal{D}}(c_{\hat{\mathbf{w}}})}{2} \right) \leq \frac{\frac{\mu^2}{2} + \ln \frac{n+1}{\delta}}{n}. \tag{7}$$

One disadvantage of the bounds in (5), (6) and (7) is that they involve a multiplicative factor of 2. In general, the factor 2 cannot be improved. However for linear classifiers with large margins there can exist tighter bounds. The following is a slightly refined version of the bounds given in [2, 3].

**Proposition 2.4** ([2, 3])**.** *Let $Q(\mu, \hat{\mathbf{w}})$ and $v_Q = c_{\hat{\mathbf{w}}}$ be defined as above. Let $er_{\mathcal{D},\theta}(Q(\mu, \hat{\mathbf{w}})) = E_{\mathbf{w} \sim \mathcal{N}(\mu\hat{\mathbf{w}}, I)} P_{\mathcal{D}} \left( y \frac{<\mathbf{w},\mathbf{x}>}{\|\mathbf{x}\|} \leq \theta \right)$ be the error of the stochastic classifier with margin $\theta$. Then for all $\theta \geq 0$*

$$er_{\mathcal{D}}(c_{\hat{\mathbf{w}}}) \leq er_{\mathcal{D},\theta}(Q(\mu, \hat{\mathbf{w}})) + \overline{\Phi}(\theta). \tag{8}$$

The bound states that if the stochastic classifier induces small errors with large margin $\theta$, then the linear (voting) classifier has only a slightly larger generalization error than the stochastic classifier. However sometimes (8) can be larger than (5). The two bounds have a different regime in which they dominate [2]. It is also worth pointing out that the margin $y\frac{<\mathbf{w},\mathbf{x}>}{\|\mathbf{x}\|}$ considered in Proposition 2.4 is unnormalized with respect to $\mathbf{w}$. See Section 3 for more discussions.

To apply Proposition 2.4, one needs to further bound $er_{\mathcal{D},\theta}(Q(\mu, \hat{\mathbf{w}}))$ by its empirical version $er_{\mathcal{S},\theta}(Q(\mu, \hat{\mathbf{w}})) := E_{\mathbf{w} \sim \mathcal{N}(\mu\hat{\mathbf{w}}, I)} P_{\mathcal{S}} \left( y \frac{<\mathbf{w},\mathbf{x}>}{\|\mathbf{x}\|} \leq \theta \right) = E_{\mathcal{S}} \overline{\Phi}(\mu y \frac{<\hat{\mathbf{w}},\mathbf{x}>}{\|\mathbf{x}\|} - \theta)$. With slight modifications of Theorem 2.2, one can show that for any $\theta \geq 0$ with probability $1-\delta$ the following bound is valid for all $\mu$ and $\hat{\mathbf{w}}$ uniformly:

$$kl \left( er_{\mathcal{S},\theta}(Q(\mu, \hat{\mathbf{w}})) \,\|\, er_{\mathcal{D},\theta}(Q(\mu, \hat{\mathbf{w}})) \right) \leq \frac{\frac{\mu^2}{2} + \ln \frac{n+1}{\delta}}{n}. \tag{9}$$

The following Proposition combines the above results.

**Proposition 2.5.** *For any $\theta \geq 0$ and any $\delta > 0$ with probability $1-\delta$ the following bound is valid for all $\mu$ and $\hat{\mathbf{w}}$ uniformly:*

$$kl \left( er_{\mathcal{S},\theta}(Q(\mu, \hat{\mathbf{w}})) \,\|\, er_{\mathcal{D}}(c_{\hat{\mathbf{w}}}) - \overline{\Phi}(\theta) \right) \leq \frac{\frac{\mu^2}{2} + \ln \frac{n+1}{\delta}}{n}. \tag{10}$$

Note that this last bound is not uniform for $\theta$, see also [3].

Improving the multiplicative factor was also studied in [22, 17], in which the variance of the stochastic classifier is also bounded by PAC-Bayes theorem, and Chebyshev inequality can be used.

## 3 Theoretical Results

In this section we give the theoretical results. The main result of this paper is Theorem 3.1, which provides a dimensionality dependent PAC-Bayes margin bound.

**Theorem 3.1.** *Let $Q(\mu, \hat{\mathbf{w}})$ ($\mu > 0$, $\hat{\mathbf{w}} \in \mathbb{R}^d$, $\|\hat{\mathbf{w}}\| = 1$) denote the distribution of linear classifiers $c_{\mathbf{w}}(\cdot) = \text{sgn}[< \mathbf{w}, \cdot >]$, where $\mathbf{w} \sim \mathcal{N}(\mu\hat{\mathbf{w}}, I)$. For any $\delta \in (0, 1)$, with probability $1-\delta$ over the random draw of $n$ training examples*

$$kl \left( er_{\mathcal{S}}(Q(\mu, \hat{\mathbf{w}})) \,\|\, er_{\mathcal{D}}(Q(\mu, \hat{\mathbf{w}})) \right) \leq \frac{\frac{d}{2} \ln(1 + \frac{\mu^2}{d}) + \ln \frac{n+1}{\delta}}{n} \tag{11}$$

*holds simultaneously for all $\mu > 0$ and all $\hat{\mathbf{w}} \in \mathbb{R}^d$ with $\|\hat{\mathbf{w}}\| = 1$. Here $er_{\mathcal{S}}(Q(\mu, \hat{\mathbf{w}})) = E_{\mathcal{S}} \overline{\Phi}(\mu\gamma(\hat{\mathbf{w}}; \mathbf{x}, y))$ and $\gamma(\hat{\mathbf{w}}; \mathbf{x}, y) = y\frac{<\hat{\mathbf{w}},\mathbf{x}>}{\|\mathbf{x}\|}$ are the same as in Theorem 2.2.*

Comparing Theorem 3.1 with Theorem 2.2, it is easy to see the following Proposition holds.

**Proposition 3.2.** *The bound (11) is sharper than (2) for any $d < \infty$, and the two bounds tend to be equivalent as $d \to \infty$.*

Theorem 3.1 is the first dimensionality dependent margin bound that remains nontrivial in infinite dimension.

Theorem 3.1 and Theorem 2.2 are uniform bounds for $\mu$. Thus one can choose appropriate $\mu$ to optimize each bound respectively. Note that $er_{\mathcal{S}}(Q(\mu, \hat{\mathbf{w}}))$ in the LHS of the two bounds is monotone decreasing with respect to $\mu$. Comparing to Theorem 2.2, Theorem 3.1 has the advantage that its RHS scales only in $O(\ln \mu)$ rather than $O(\mu^2)$, and therefore allows choosing a very large $\mu$.

As described in (7) in Section 2, we can also obtain a margin bound for the deterministic linear classifier $c_{\hat{\mathbf{w}}}$ by combining (11) with $er_{\mathcal{D}}(c_{\hat{\mathbf{w}}}) \leq 2\, er_{\mathcal{D}}(Q(\mu, \hat{\mathbf{w}}))$.

In addition, note that the VC dimension of homogeneous linear classifiers in $\mathbb{R}^d$ is $d$. From Theorem 3.1 we can almost recover the VC bound [23]

$$er_{\mathcal{D}}(c) \leq er_{\mathcal{S}}(c) + \sqrt{\frac{d\left(1 + \ln\left(\frac{2n}{d}\right)\right) + \ln\frac{4}{\delta}}{n}} \tag{12}$$

for homogenous linear classifiers in $\mathbb{R}^d$ under mild conditions. Formally we have the following Corollary.

**Corollary 3.3.** *Theorem 3.1 implies the following result. Suppose $n > 5$. For any $\delta > 2e^{-\frac{d}{8}}n^{-\frac{1}{8}}$, with probability $1 - \delta$ over the random draw of $n$ training examples*

$$er_{\mathcal{D}}(c_{\boldsymbol{w}}) \leq er_{\mathcal{S}}(c_{\boldsymbol{w}}) + \sqrt{\frac{d\ln\left(1 + \left(\frac{2n}{d}\right)\right) + \frac{1}{2}\ln\frac{2(n+1)}{\delta}}{n}} + \sqrt{\frac{d + \ln n}{n}} \tag{13}$$

*holds simultaneously for all homogeneous linear classifiers $c_{\boldsymbol{w}}$ with $\boldsymbol{w} \in \mathbb{R}^d$ satisfying*

$$P_{\mathcal{D}}\left(\left|y\frac{<\boldsymbol{w}, \boldsymbol{x}>}{\|\boldsymbol{w}\|\|\boldsymbol{x}\|}\right| \leq \frac{(\ln n)^{1/2}d^{3/2}}{4n^2}\right) \leq \frac{1}{4}\sqrt{\frac{d + \ln n}{n}}. \tag{14}$$

Condition (14) is easy to satisfy if $d \ll n$.

In a sense, the dimensionality dependent margin bound in Theorem 3.1 unifies the dimensionality independent margin bound and the VC bound for linear classifiers.

Although it is not easy to theoretically quantify how much sharper (11) is than (2) and the VC bound (12) (because the first two bounds hold uniformly for all $\mu$), in Section 4 we will demonstrate by experiments that the new bound is usually significantly better than (2) and (12) on relatively large datasets.

### 3.1 Improving the Multiplicative Factor

As we mentioned in Section 2, Proposition 2.3 involves a multiplicative factor of 2 when bounding the error of the deterministic voting classifier by the error of the stochastic classifier. Note that in general $er_{\mathcal{D}}(c_{\hat{\mathbf{w}}}) \leq 2er_{\mathcal{D}}(Q(\mu, \hat{\mathbf{w}}))$ cannot be improved (consider the case that with probability one the data has zero margin with respect to $\hat{\mathbf{w}}$). Here we study how to improve it for large margin classifiers.

Recall that Proposition 2.4 gives $er_{\mathcal{D}}(c_{\hat{\mathbf{w}}}) \leq er_{\mathcal{D}, \theta}(Q(\mu, \hat{\mathbf{w}})) + \overline{\Phi}(\theta)$, which bounds the generalization error of the linear classifier in terms of the error of the stochastic classifier with margin $\theta \geq 0$. As pointed out in [2], this bound is not always better than Proposition 2.3 (i.e., $er_{\mathcal{D}}(c_{\hat{\mathbf{w}}}) \leq 2er_{\mathcal{D}}(Q(\mu, \hat{\mathbf{w}}))$). The two bounds each has a different dominant regime. Our first result in this subsection is the following simple improvement over both Proposition 2.3 and Proposition 2.4.

**Proposition 3.4.** *Using the notions in Proposition 2.4, we have that for all $\theta \geq 0$,*

$$er_{\mathcal{D}}(c_{\hat{\mathbf{w}}}) \leq \frac{1}{\Phi(\theta)}er_{\mathcal{D}, \theta}(Q(\mu, \hat{\mathbf{w}})), \tag{15}$$

*where $\Phi(\theta)$ is defined in Theorem 2.2.*

It is easy to see that Proposition 2.3 is a special case of Proposition 3.4: just let $\theta = 0$ in (15) we recover (6). Thus Proposition 3.4 is always sharper than Proposition 2.3. It is also easy to show that (15) is sharper than (8) in Proposition 2.4 whenever the bounds are nontrivial. Formally we have the following proposition.

**Proposition 3.5.** *Suppose the RHS of (8) or the RHS of (15) is smaller than* 1, *i.e., at least one of the two bounds is nontrivial. Then (15) is sharper than (8).*

As mentioned in Section 2, the margins discussed so far in this subsection are unnormalized with respect to $\mathbf{w} \in \mathbb{R}^d$. That is, we consider $y\frac{<\mathbf{w},\mathbf{x}>}{\|\mathbf{x}\|}$. In the following we will focus on normalized margins $y\frac{<\mathbf{w},\mathbf{x}>}{\|\mathbf{w}\|\|\mathbf{x}\|}$. It will soon be clear that this brings additional benefits when combining with the dimensionality dependent margin bound.

Let $er^{\mathbf{N}}_{\mathcal{D},\theta}(Q(\mu,\hat{\mathbf{w}})) = E_{w \sim \mathcal{N}(\mu\hat{\mathbf{w}},I)} P_{\mathcal{D}}(y\frac{<\mathbf{w},\mathbf{x}>}{\|\mathbf{w}\|\|\mathbf{x}\|} \leq \theta)$ be the true error of the stochastic classifier $Q(\mu,\hat{\mathbf{w}})$ with normalized margin $\theta \in [-1,1]$. Also let $er^{\mathbf{N}}_{\mathcal{S},\theta}(Q(\mu,\hat{\mathbf{w}}))$ be its empirical version. We have the following lemma.

**Lemma 3.6.** *For any $\mu > 0$, any $\hat{\boldsymbol{w}} \in \mathbb{R}^d$ with $\|\hat{\boldsymbol{w}}\| = 1$ and any $\theta \geq 0$,*

$$er_{\mathcal{D}}(c_{\hat{\boldsymbol{w}}}) \leq \frac{er^{\mathbf{N}}_{\mathcal{D},\theta}(Q(\mu,\hat{\boldsymbol{w}}))}{\Phi(\mu\theta)}. \tag{16}$$

If $er^{\mathbf{N}}_{\mathcal{D},\theta}(Q)$ is only slightly larger than $er_{\mathcal{D}}(Q)$ for a not-too-small $\theta > 0$, then $\frac{er^{\mathbf{N}}_{\mathcal{D},\theta}(Q)}{\Phi(\mu\theta)}$ can be much smaller than $2er_{\mathcal{D}}(Q)$ even with a not too large $\mu$. Also note that setting $\theta = 0$ in (16), we can recover (6).

The true margin error $er^{\mathbf{N}}_{\mathcal{D},\theta}(Q)$ can be bounded by its empirical version similar to Theorem 3.1: For any $\theta \geq 0$ and any $\delta > 0$, with probability $1 - \delta$

$$kl\left(er^{\mathbf{N}}_{\mathcal{S},\theta}(Q(\mu,\hat{\mathbf{w}}))\|er^{\mathbf{N}}_{\mathcal{D},\theta}(Q(\mu,\hat{\mathbf{w}}))\right) \leq \frac{\frac{d}{2}\ln(1 + \frac{\mu^2}{d}) + \ln\frac{n+1}{\delta}}{n} \tag{17}$$

holds simultaneously for all $\mu > 0$ and $\hat{\mathbf{w}} \in \mathbb{R}^d$ with $\|\hat{\mathbf{w}}\| = 1$.

Combining the previous two results we have a dimensionality dependent margin bound for the linear classifier $c_{\hat{\mathbf{w}}}$.

**Proposition 3.7.** *Let $Q(\mu,\hat{\boldsymbol{w}})$ defined as before. For any $\theta \geq 0$ and any $\delta > 0$, with probability $1 - \delta$ over the random draw of $n$ training examples*

$$kl\left(er^{\mathbf{N}}_{\mathcal{S},\theta}(Q(\mu,\hat{\boldsymbol{w}}))\|er_{\mathcal{D}}(c_{\hat{\boldsymbol{w}}})\Phi(\mu\theta)\right) \leq \frac{\frac{d}{2}\ln(1 + \frac{\mu^2}{d}) + \ln\frac{n+1}{\delta}}{n} \tag{18}$$

*holds simultaneously for all $\mu > 0$ and $\hat{\boldsymbol{w}} \in \mathbb{R}^d$ with $\|\hat{\boldsymbol{w}}\| = 1$.*

To see how Proposition 3.7 improves the multiplicative factor, let's take a closer look at the bound (18). Observe that as $\mu$ getting large, $er^{\mathbf{N}}_{\mathcal{S},\theta}(Q(\mu,\hat{\mathbf{w}})) = E_{\mathbf{w} \sim \mathcal{N}(\mu\hat{\mathbf{w}},I)} P_{\mathcal{D}}(y\frac{<\mathbf{w},\mathbf{x}>}{\|\mathbf{w}\|\|\mathbf{x}\|} \leq \theta)$ tends to the empirical error of the linear classifier $\hat{\mathbf{w}}$ with margin $\theta$, i.e., $P_{\mathcal{S}}\left(y\frac{<\hat{\mathbf{w}},\mathbf{x}>}{\|\mathbf{x}\|} \leq \theta\right)$ (recall that $\|\hat{\mathbf{w}}\|=1$). Also if $\mu\theta > 3$, $\Phi(\mu\theta) \approx 1$. Taking into the consideration that the RHS of (18) scales only in $O(\ln \mu)$, we can choose a relatively large $\mu$ and (18) gives a dimensionality dependent margin bound whose multiplicative factor can be very close to 1.

## 4  Experiments

In this section we conduct a series of experiments on benchmark datasets. The goal is to see to what extent the Dimensionality Dependent margin bound (will be referred to as DD-margin bound) is sharper than the Dimensionality Independent margin bound (will be referred to as DI-margin bound) as well as the VC bound. More importantly, we want to see from the experiments how useful the DD-margin bound is for model selection.

Table 1: Description of dataset

| Dataset | # Examples | # Features | Dataset | # Examples | # Features |
|---|---|---|---|---|---|
| Image | 2310 | 20 | Letter | 20000 | 16 |
| Magic04 | 19020 | 10 | Mushroom | 8124 | 22 |
| Optdigits | 5620 | 64 | PageBlock | 5473 | 10 |
| Pendigits | 10992 | 16 | Waveform | 3304 | 21 |
| BreastCancer | 683 | 9 | Glass | 214 | 9 |
| Pima | 768 | 8 | wdbc | 569 | 30 |

We use 12 datasets all from the UCI repository [24]. A description of the datasets is given in Table 1. For each dataset, we use 5-fold cross validation and average the results over 10 runs (for a total 50 runs). If the dataset is a multiclass problem, we group the data into two classes since we study binary classification problems. In the data preprocessing stage each feature is normalized to $[0, 1]$.

To compare the bounds and to do model selection, we use SVM with polynomial kernels $K(\mathbf{x}, \mathbf{x}') = (a < \mathbf{x}, \mathbf{x}' > +b)^t$ and let $t$ varies[4]. For each $t$, we train a classifier by libsvm [25]. We plot the values of the three bounds—the DD-margin bound, the DI-margin bound, the VC bound (12) as well as the test and training error (see Figure 1 - Figure 12). For the two margin bounds, since they hold uniformly for $\mu > 0$, we select the optimal $\mu$ to make the bounds as small as possible. For simplicity, we combine Proposition 2.3 with Theorem 3.1 and Theorem 2.2 respectively to obtain the final bound for the generalization error of the deterministic linear classifiers. In each figure, the horizonal axis represents the degree $t$ of the polynomial kernel. All bounds in the figures (including training and test error) are for deterministic (voting) classifier.

To analyze the experimental results, we group the 12 results into two categories as follows.

1. Figure 1 - Figure 8. This category consists of eight datasets, and each of them contains at least 2000 examples (relatively large datasets). On all these datasets, the DD-margin bounds are significantly sharper than the DI-margin bounds as well as the VC bounds. More importantly, the DD-margin bounds work well for model selection. We can use this bound to choose the degree of the polynomial kernel. On all the datasets except "Image", the curve of the DD-margin bound is highly correlated with the curve of the test error: When the test error decreases (or increases), the DD-margin bound also decreases (or increases); And as the test error remains unchanged as the degree $t$ grows, the DD-margin bound selects the model with the lowest complexity.

2. Figure 9 - Figure 12. This category consists of four small datasets, each contains less than 1000 examples. On these small datasets, the VC bounds often become trivial (larger than 1). The DD-margin bounds are still always, but less significantly, sharper than the DI-margin bounds. However, on these small datasets, it is difficult to tell if the bounds select good models.

In sum, the experimental results demonstrate that the DD-margin bound is usually significantly sharper than the DI-margin bound as well as the VC bound if the dataset is relatively large. Also the DD-margin bound is useful for model selection. However, for small datasets, all three bounds seem not useful for practical purpose.

## 5 Conclusion

In this paper we study the problem whether dimensionality independency is intrinsic for margin bounds. We prove a dimensionality dependent PAC-Bayes margin bound. This bound is sharper than a previously well-known dimensionality independent margin bound when the feature space is of finite dimension; and they tend to be equivalent as the dimensionality grows to infinity. Experimental results demonstrate that for relatively large datasets the new bound is often useful for model selection and significantly sharper than previous margin bound as well as the VC bound.

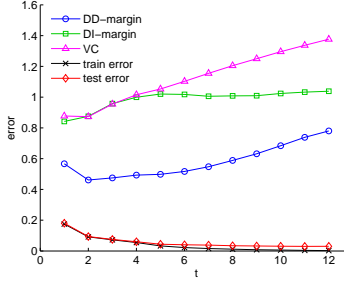

Figure 1: Image

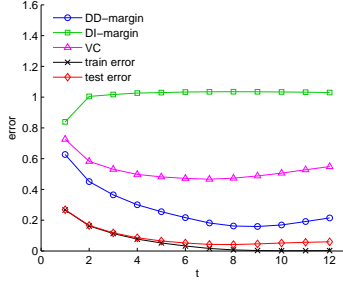

Figure 2: Letter

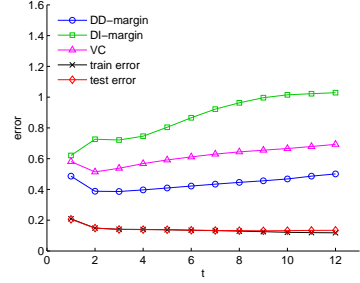

Figure 3: Magic04

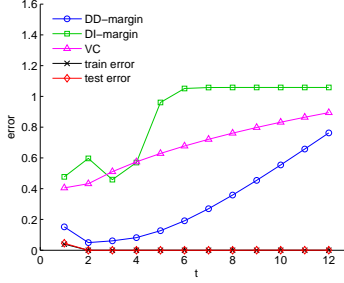

Figure 4: Mushroom

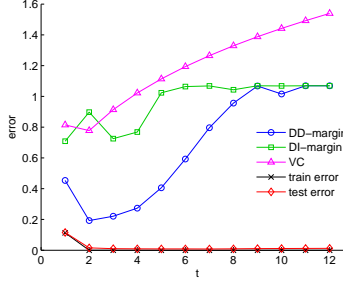

Figure 5: Optdigits

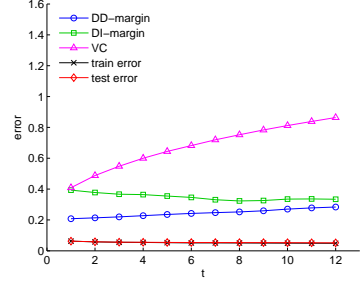

Figure 6: PageBlocks

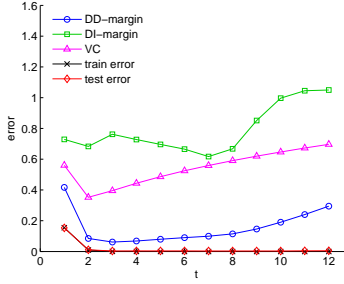

Figure 7: Pendigits

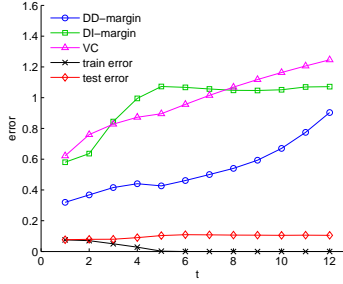

Figure 8: Waveform

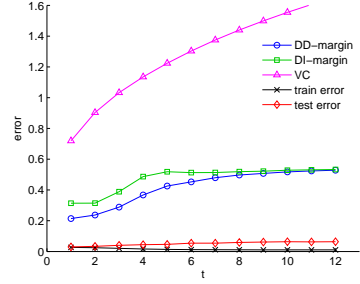

Figure 9: BreastCancer

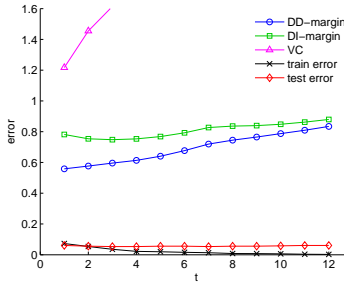

Figure 10: Glass

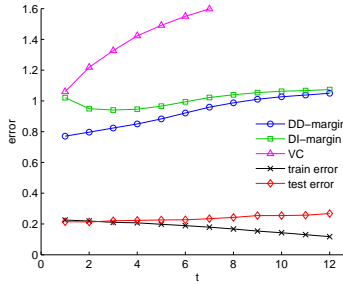

Figure 11: Pima

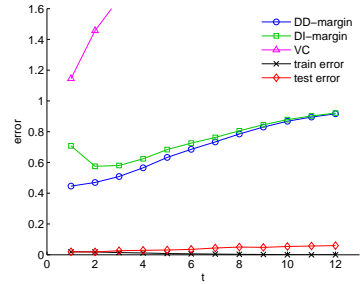

Figure 12: wdbc

Our work is based on the PAC-Bayes theory. One limitation is that it involves a multiplicative factor of 2 when transforming stochastic classifiers to deterministic classifiers. Although we provide two improved bounds (Proposition 3.4, 3.7) over previous results (Proposition 2.3, 2.4), the multiplicative factor is still strictly larger than 1. A future work is to study whether there exist dimensionality dependent margin bounds (not necessarily PAC-Bayes) without this multiplicative factor.

### Acknowledgments

This work was supported by NSFC(61222307, 61075003) and a grant from Microsoft Research Asia. We also thank Chicheng Zhang for very helpful discussions.

## Footnotes

[1] The bound depends on the VC dimension of the base hypothesis class. Nevertheless, given the VC dimension of the base hypothesis space, the bound does not depend on the number of the base classifiers, which can be seen as the dimension of the feature space.

[2]This does not sacrifice any generality since linear classifiers can be easily transformed to homogeneous linear classifiers by adding a new dimension.

[3]There exist dimensionality dependent margin bounds [21]. However these bounds grow unboundedly as the dimensionality tends to infinity.

[4] For simplicity we fix $a$ and $b$ as constants in all the experiments.

# References

[1] John Shawe-Taylor, Peter L. Bartlett, Robert C. Williamson, and Martin Anthony. Structural risk minimization over data-dependent hierarchies. *IEEE Transactions on Information Theory*, 44(5):1926–1940, 1998.

[2] John Langford and John Shawe-Taylor. PAC-Bayes & Margins. In *Advances in Neural Information Processing Systems*, pages 423–430, 2002.

[3] David A. McAllester. Simplified PAC-Bayesian margin bounds. *Learning Theory and Kernel Machines*, 2777:203–215, 2003.

[4] John Langford. Tutorial on practical prediction theory for classification. *Journal of Machine Learning Research*, 6:273–306, 2005.

[5] Matthias Seeger. PAC-Bayesian generalization error bounds for Gaussian process classification. *Journal of Machine Learning Research*, 3:233–269, 2002.

[6] David A. McAllester. Some PAC-Bayesian theorems. *Machine Learning*, 37(3):355–363, 1999.

[7] Robert E. Schapire, Yoav Freund, Peter Barlett, and Wee Sun Lee. Boosting the margin: A new explanation for the effectiveness of voting methods. *Annals of Statistics*, 26(5):1651–1686, 1998.

[8] Vladimir Koltchinskii and Dmitry Panchenko. Empirical margin distributions and bounding the generalization error of combined classifiers. *Annals of Statistics*, 30:1–50, 2002.

[9] Vladimir Koltchinskii and Dmitry Panchenko. Complexities of convex combinations and bounding the generalization error in classification. *Annals of Statistics*, 33:1455–1496, 2005.

[10] John Langford, Matthias Seeger, and Nimrod Megiddo. An improved predictive accuracy bound for averaging classifiers. In *International Conference on Machine Learning*, pages 290–297, 2001.

[11] Sanjoy Dasgupta and Philip M. Long. Boosting with diverse base classifiers. In *Annual Conference on Learning Theory*, pages 273–287, 2003.

[12] Leo Breiman. Prediction games and arcing algorithms. *Neural Computation*, 11:1493–1518, 1999.

[13] Liwei Wang, Masashi Sugiyama, Zhaoxiang Jing, Cheng Yang, Zhi-Hua Zhou, and Jufu Feng. A refined margin analysis for boosting algorithms via equilibrium margin. *Journal of Machine Learning Research*, 12:1835–1863, 2011.

[14] Olivier Catoni. PAC-Bayesian supervised classification: The thermodynamics of statistical learning. *IMS Lecture Notes–Monograph Series*, 56, 2007.

[15] Pascal Germain, Alexandre Lacasse, François Laviolette, and Mario Marchand. PAC-Bayesian learning of linear classifiers. In *International Conference on Machine Learning*, page 45, 2009.

[16] Pascal Germain, Alexandre Lacasse, François Laviolette, Mario Marchand, and Sara Shanian. From PAC-Bayes bounds to KL regularization. In *Advances in Neural Information Processing Systems*, pages 603–610, 2009.

[17] Jean-Francis Roy, François Laviolette, and Mario Marchand. From PAC-Bayes bounds to quadratic programs for majority votes. In *International Conference on Machine Learning*, pages 649–656, 2011.

[18] Amiran Ambroladze, Emilio Parrado-Hernández, and John Shawe-Taylor. Tighter pac-bayes bounds. In *Advances in Neural Information Processing Systems*, pages 9–16, 2006.

[19] John Shawe-Taylor, Emilio Parrado-Hernández, and Amiran Ambroladze. Data dependent priors in PAC-Bayes bounds. In *International Conference on Computational Statistics*, pages 231–240, 2010.

[20] Peter L. Bartlett. The sample complexity of pattern classification with neural networks: the size of the weights is more important than the size of the network. *IEEE Transactions on Information Theory*, 44(2):525–536, 1998.

[21] Ralf Herbrich and Thore Graepel. A PAC-Bayesian margin bound for linear classifiers. *IEEE Transactions on Information Theory*, 48(12):3140–3150, 2002.

[22] Alexandre Lacasse, François Laviolette, Mario Marchand, Pascal Germain, and Nicolas Usunier. PAC-Bayes bounds for the risk of the majority vote and the variance of the gibbs classifier. In *Advances in Neural Information Processing Systems*, pages 769–776, 2006.

[23] Vladimir N. Vapnik. *Statistical Learning Theory*. Wiley-Interscience, 1998.

[24] Andrew Frank and Arthur Asuncion. UCI machine learning repository, 2010.

[25] Chih-Chung Chang and Chih-Jen Lin. LIBSVM: A library for support vector machines. *ACM Transactions on Intelligent Systems and Technology*, 2:27:1–27:27, 2011.

